# Efficient Exact k-NN and Nonparametric Classification in High Dimensions

**Ting Liu**
Computer Science Dept.
Carnegie Mellon University
Pittsburgh, PA 15213
tingliu@cs.cmu.edu

**Andrew W. Moore**
Computer Science Dept.
Carnegie Mellon University
Pittsburgh, PA 15213
awm@cs.cmu.edu

**Alexander Gray**
Computer Science Dept.
Carnegie Mellon University
Pittsburgh, PA 15213
agray@cs.cmu.edu

## Abstract

This paper is about non-approximate acceleration of high dimensional nonparametric operations such as $k$ nearest neighbor classifiers and the prediction phase of Support Vector Machine classifiers. We attempt to exploit the fact that even if we want exact answers to nonparametric queries, we usually do not need to explicitly find the datapoints close to the query, but merely need to ask questions about the properties about that set of datapoints. This offers a small amount of computational leeway, and we investigate how much that leeway can be exploited. For clarity, this paper concentrates on pure $k$-NN classification and the prediction phase of SVMs. We introduce new ball tree algorithms that on real-world datasets give accelerations of 2-fold up to 100-fold compared against highly optimized traditional ball-tree-based $k$-NN. These results include datasets with up to $10^6$ dimensions and $10^5$ records, and show non-trivial speedups while giving exact answers.

## 1 Introduction

Nonparametric models have become increasingly popular in the statistics communities and probabilistic AI communities. They remain hampered by their computational complexity. Spatial methods such as kd-trees [6, 17], R-trees [9], metric trees [18, 4] and ball trees [15] have been proposed and tested as a way of alleviating the computational cost of such statistics without resorting to approximate answers. They have been used in many different ways, and with a variety of tree search algorithms and with a variety of "cached sufficient statistics" decorating the internal leaves, for example in [14, 5, 16, 8].

The main concern with such accelerations is the extent to which they can survive high dimensional data. Indeed, there are some datasets in this paper for which a highly optimized conventional $k$ nearest neighbor search based on ball trees is on average *more* expensive than the naive linear search algorithm,but extracting the $k$ nearest neighbors is often not needed, even for a $k$ nearest neighbor classifier. This paper is about the consequences of the fact that none of these three questions have the same precise meaning: (a) *"What are the k nearest neighbors of* **t***?"* (b) *"How many of the k nearest neighbors of* **t** *are from the positive class?"* and (c) *"Are at least q of the k nearest neighbors from the positive class?"* The computational geometry community has focused on question (a), but uses of proximity queries in statistics far more frequently require (b) and (c) types of computations. Further, in addition to traditional $K$-NN, the same insight applies to many other statistical computations such as nonparametric density estimation, locally weighted regression, mixture models, $k$-means and the prediction phase of SVM classification.

## 2 Ball trees

A *ball tree* is a binary tree in which each node represents a set of points, called Points(Node). Given a dataset, the *root node* of a ball tree represents the full set of points in the dataset. A node can be either a *leaf node* or a *non-leaf node*. A leaf node explicitly contains a list of the points represented by the node. A non-leaf node does not explicitly contain a set of points. It has two child nodes: *Node.child1* and *Node.child2*, where

$$Points(Node.child1) \cap Points(Node.child2) = \phi$$

$$Points(Node.child1) \cup Points(Node.child2) = Points(Node)$$

Points are organized spatially. Each node has a distinguished point called a *pivot*. Depending on the implementation, the pivot may be one of the datapoints, or it may be the centroid of *Points(Node)*. Each node records the maximum distance of the points it owns to its pivot. Call this the radius of the node

$$Node.Radius = max_{\mathbf{x} \in Points(Node)} \,| \, Node.Pivot - \mathbf{x} \,|$$

Balls lower down the tree cover smaller volumes. This is achieved by insisting, at tree construction time, that

$$\mathbf{x} \in Points(Node.child1) \quad \Rightarrow \quad |\, \mathbf{x} - Node.child1.Pivot \,| \;\leq\; |\, \mathbf{x} - Node.child2.Pivot \,|$$

$$\mathbf{x} \in Points(Node.child2) \quad \Rightarrow \quad |\, \mathbf{x} - Node.child2.Pivot \,| \;\leq\; |\, \mathbf{x} - Node.child1.Pivot \,|$$

Provided our distance function obeys the triangle inequality, this gives the ability to bound the distance from a target point $\mathbf{t}$ to any point in any ball tree node. If $\mathbf{x} \in Points(Node)$ then we can be sure that:

$$|\mathbf{x} - \mathbf{t}| \quad \geq \quad |\mathbf{t} - Node.\mathbf{Pivot}| - Node.Radius \tag{1}$$

$$|\mathbf{x} - \mathbf{t}| \quad \leq \quad |\mathbf{t} - Node.\mathbf{Pivot}| + Node.Radius \tag{2}$$

Ball trees are constructed top-down. There are several ways to construct them, and practical algorithms trade off the cost of construction (it would be useless to be $O(R^2)$ given a dataset with $R$ points, for example) against the tightness of the radius of the balls. [13] describes one fast way of constructing a ball tree appropriate for computational statistics. If a ball tree is balanced, then the construction time is $O(CR \log R)$, where $C$ is the cost of a point-point distance computation (which is $O(m)$ if there are $m$ dense attributes, and $O(fm)$ if the records are sparse with only fraction $f$ of attributes taking non-zero values).

### 2.1 KNS1: Conventional $K$ nearest neighbor search with ball trees

In this paper, we call conventional ball-tree-based search [18] *KNS1*. Let a pointset *PS* be a set of datapoints. We begin with the following definition:
Say that *PS* consists of the $k$-NN of $\mathbf{t}$ in pointset $V$ if and only if

$$((|\,V\,| \geq k) \wedge (PS \text{ are the } k\text{-NN of } \mathbf{t} \text{ in } V)) \vee ((|\,V\,| < k) \wedge (PS = V)) \tag{3}$$

We now define a recursive procedure called *BallKNN* with the following inputs and output.

$$PS^{out} = BallKNN(PS^{in}, Node)$$

Let $V$ = set of points searched so far, on entry. Assume $PS^{in}$ consists of the $k$-NN of $\mathbf{t}$ in V. This function efficiently ensures that on exit, $PS^{out}$ consists of the $k$-NN of $\mathbf{t}$ in $V \cup Points(Node)$.

$$\text{Let } D_{\text{sofar}} = \begin{cases} \infty & if \ |\,PS^{in}\,| < k \\ max_{\mathbf{x} \in PS^{in}} |\,\mathbf{x} - \mathbf{t}\,| & if \ |\,PS^{in}\,| = k \end{cases} \tag{4}$$

$D_{\text{sofar}}$ is the minimum distance within which points would become interesting to us.

$$\text{Let } D_{\text{minp}}^{\text{Node}} = \begin{cases} max(|\mathbf{t} - Node.\mathbf{Pivot}| - Node.Radius, D_{minp}^{Node.parent}) & if \ Node \neq Root \\ max(|\mathbf{t} - Node.\mathbf{Pivot}| - Node.Radius, 0) & if \ Node = Root \end{cases} \tag{5}$$

$D_{\text{minp}}^{\text{Node}}$ is the minimum possible distance from any point in *Node* to $\mathbf{t}$.

```
Procedure BallKNN (PS^in, Node)
begin
  if (D_minp^Node ≥ D_sofar) then exit returning PS^in unchanged.
  else if (Node is a leaf) PS^out = PS^in
  ∀x ∈ Points(Node)
    if (| x − t | < D_sofar) then
      add x to PS^out
      if (| PS^out | = k + 1) then
        remove furthest neighbor from PS^out; update D_sofar
  else if (Node is a non-leaf)
    node_1 = child of Node closest to t
    node_2 = child of Node furthest from t
    PS^temp = BallKNN(PS^in, node_1)
    PS^out = BallKNN(PS^temp, node_2)
end
```

A call of BallKNN({},Root) returns the $k$ nearest neighbors of **t** in the Ball tree.

## 2.2 KNS2: Faster $k$-NN classification for skewed-class data

In several binary classification domains,one class is much more frequent than the other, For example, in High Throughput Screening datasets, [19] it is far more common for the result of an experiment to be negative than positive. In fraud detection or intrusion detection, a non-attack is far more common than an attack. The new algorithm introduced in this section, KNS2, is designed to accelerate $k$-NN based classification beyond the speedups already available by using KNS1 (conventional ball-tree-based $k$-NN). KNS2 attacks the problem by building two ball trees: $Root_{pos}$ is the root of a (small) ball tree built from all the positive points in the dataset. $Root_{neg}$ is the root of a (large) ball tree built from all negative points.

Then, when it is time to classify a new target point **t**, we compute $q$, the number of $k$ nearest neighbors of **t** that are in the positive class, in the following fashion

- Step 1 — " **Find positive**": Find the $k$ nearest positive class neighbors of **t** (and their distances to **t**) using conventional ball tree search.

- Step 2 — **"Insert negative"**: Do sufficient search of the negative tree to prove that the number of positive datapoints among $k$ nearest neighbors is $q$ for some value of $q$.

Step 2 is achieved using a new recursive search called *NegCount*. In order to describe *NegCount* we need the following three definitions.

- **The Dists Array.** *Dists* is an array of elements $Dists_1 \ldots Dists_k$ consisting of the distances to the $k$ nearest positive neighbors of **t**, sorted in increasing order of distance. We will also write $Dists_0 = 0$ and $Dists_{k+1} = \infty$.

- **Pointsets.** Define pointset $V$ as the set of points in the negative balls visited so far.

- **The Counts Array (n,C).** Say that *(n,C) summarize interesting negative points for pointset V* if and only if

  1. $\forall i \in [0, n]$,
$$C_i = | V \cap \{x : Dists_i \leq | x - t | < Dists_{i+1}\} | \qquad (6)$$

  2. $\sum_{i=0}^{n} C_i \geq k$, $\sum_{i=0}^{n-1} C_i < k$. This simply declares that the length $n$ of the $C$ array is as short as possible while accounting for the $k$ members of $V$ that are nearest to **t**.

Step 2 of KNS2 is implemented by the recursive function

$$(n^{out}, C^{out}) = NegCount(n^{in}, C^{in}, Node, Dists)$$

Assume that on entry that $(n^{in}, C^{in})$ summarize interesting negative points for pointset $V$, where $V$ is the set of points visited so far during the search. This algorithm efficiently ensures that on exit $(n^{out}, C^{out})$ summarize interesting negative points for $V \cup Points(Node)$.

---

**Procedure** NegCount $(n^{in}, C^{in}, Node, Dists)$
**begin**
  $n^{out} \ := \ n^{in}; C^{out} \ := \ C^{in}$
  Let $T = \sum_{i=0}^{n^{in}-1} C_i^{in}$
  T is the total number of negative points closer than the $n^{in}$th positive point

  **if** $(D_{\min p}^{Node} \geq Dist_{n^{in}})$ **then** exit and return$(n^{out}, C^{out})$
  **else if** (Node is a leaf)
    $\forall \mathbf{x} \in Points(Node)$
    Use binary search to find $j \in [0, n^{out}]$, such that $Dists_j \leq |\mathbf{x} - \mathbf{t}| < Dists_{j+1}$
    $C_j^{out} \ := \ C_j^{out} + 1; T \ := \ T + 1$
    If $T$ exceeds $k$, decrement $n^{out}$ until $T = \sum_{i=0}^{n^{out}-1} C_i^{out} < k$.
    $Dists_{n^{out}+1} \ := \ \infty$
    **if** $(n^{out} = 0)$exit and return$(0, C^{out})$
  **else if**(Node is a non leaf)
    $node_1 \ := \$ child of Node closest to $\mathbf{t}$
    $node_2 \ := \$ child of Node furthest from $\mathbf{t}$
    $(n^{temp}, C^{temp}) \ := \$ NegCount$(n^{in}, C^{in}, node_1, Dists)$
    **if** $(n^{temp} = 0)$ exit and return $(0, C^{out})$
    $(n^{out}, C^{out}) \ := \$ NegCount$(n^{temp}, C^{temp}, node_2, Dists)$
**end**

---

We can stop the procedure when $n^{out}$ becomes 0 (which means all the $k$ nearest neighbors of $\mathbf{t}$ are in the negative class) or when we run out of nodes. The top-level call is

$$NegCount(k, C^0, NegTree.Root, Dists)$$

where $C^0$ is an array of zeroes and $Dists$ are defined in Equation 6 and obtained by applying KNS1 to the (small) positive ball tree.

### 2.3 KNS3: Are at least $q$ of the $k$ nearest neighbors positive?
Unfortunately, space constraints prevent us from describing the details of KNS3. KNS3 removes KNS2's constraint of an assumed skewedness in the class distribution, while introducing a new constraint: we answer the binary question "are at least $q$ nearest neighbors positive?" (where the questioner must supply $q$). This is often the most statistically relevant question, for example during classification with known false positive and false negative costs. KNS3 will be described fully in a journal-article length version of the paper [1].

### 2.4 SVP1: Faster Radial Basis SVM Prediction
After an SVM [3] has been trained we hit the prediction phase. Given a batch of query points $\mathbf{q}_1, \mathbf{q}_2 \ldots \mathbf{q}_R$ we wish to classify each $\mathbf{q}_j$. Furthermore, in state-of-the-art training algorithms such as SMO, training time is dominated by SVM evaluation [12]. $\mathbf{q}_j$ should be classified according to this rule:

$$ASUM(\mathbf{q}_j) = \sum_{i \in \mathbf{posvecs}} \alpha_i K(|\mathbf{q}_j - \mathbf{x}_i|) \ , \ BSUM(\mathbf{q}_j) = \sum_{i \in \mathbf{negvecs}} \beta_i K(|\mathbf{q}_j - \mathbf{x}_i|) \qquad (7)$$

[1]available from www.autonlab.org

$$Class(\mathbf{q}_j) \quad = 1 \quad \text{if } ASUM(\mathbf{q}_j) - BSUM(\mathbf{q}_j) \geq -b$$
$$= 0 \quad \text{if } ASUM(\mathbf{q}_j) - BSUM(\mathbf{q}_j) < -b$$

Where the positive support vectors **posvecs**, the negative support vectors **negvecs** and the weights $\{\alpha_i\}$, $\{\beta_i\}$ and constant term $b$ are all obtained from SVM training.

We place the *queries* (not the support vectors) into a ball-tree. We can then apply the same kinds of tricks as KNS2 and KNS3 in which we do not need to find the explicit values of the *ASUM* and *BSUM* terms, but merely find balls in the tree in which we can prove all query points satisfy one of the above inequalities.

To classify all the points in a node called *Node* we do the following:

1. Compute values $(ASUM^{\text{LO}}, ASUM^{\text{HI}})$ such that we can be sure

$$\forall \mathbf{q}_j \in Node : ASUM^{\text{LO}} \leq ASUM(\mathbf{q}_j) \leq ASUM^{\text{HI}} \tag{8}$$

without iterating over the queries in *Node*. This is achieved simply, for example if $\mathbf{q}_j \in Node$ we know

$$ASUM(\mathbf{q}_j) \quad = \quad \sum_{i \in \textbf{posvecs}} \alpha_i K(|\mathbf{q}_j - \mathbf{x}_i|)$$
$$\geq \quad \sum_{i \in \textbf{posvecs}} \alpha_i K(|Node.pivot - \mathbf{x}_i| + Node.Radius)$$
$$= \quad ASUM^{\text{LO}}$$

Similarly,

$$ASUM(\mathbf{q}_j) \quad = \quad \sum_{i \in \textbf{posvecs}} \alpha_i K(|\mathbf{q}_j - \mathbf{x}_i|)$$
$$\leq \quad \sum_{i \in \textbf{posvecs}} \alpha_i K(max(|Node.pivot - \mathbf{x}_i| - Node.Radius, 0))$$
$$= \quad ASUM^{\text{HI}}$$

under the assumption that the kernel function is a decreasing function of distance. This is true, for example, for Gaussian Radial Basis function kernels.

2. Similarly compute values $(BSUM^{\text{LO}}, BSUM^{\text{HI}})$.

3. If $ASUM^{\text{LO}} - BSUM^{\text{HI}} \geq -b$ we have proved that all queries in *Node* should be classified positively, and we can terminate this recursive call.

4. If $ASUM^{\text{HI}} - BSUM^{\text{LO}} < -b$ we have proved that all queries in *Node* should be classified negatively, and we can terminate this recursive call.

5. Else we recurse and apply the same procedure to the two children of *Node*, unless *Node* is a leaf node in which case we must explicitly iterate over its members.

## 3 Experimental Results

Table 1 is a summary of the datasets in the empirical analysis.

**Life Sciences:** These were proprietary datasets (*ds1* and *ds2*) similar to the publicly available Open Compound Database provided by the National Cancer Institute (NCI Open Compound Database, 2000). The two datasets are sparse. We also present results on datasets derived from *ds1*, denoted *ds1.10pca*, *ds1.100pca* and *ds2.100anchor* by linear projection using principal component analysis (PCA).

**Link Detection:** The first, Citeseer, is derived from the Citeseer web site (Citeseer,2002) and lists the names of collaborators on published materials. The goal is to predict whether J_Lee ( the most common name) was a collaborator for each work based on who else is

listed for that work. We use *J_Lee.100pca* to represent the linear projection of the data to 100 dimensions using PCA. The second link detection dataset is derived from the Internet Movie Database (IMDB,2002) and is denoted *imdb* using a similar approach, but to predict the participation of Mel Blanc (again the most common participant).

**UCI/KDD data:** We use three large datasets from KDD/UCI repository [2]. The datasets can be identified from their names. They were converted to binary classification problems. Each categorical input attribute was converted into *n* binary attributes by a 1-of-*n* encoding (where *n* is the attribute's arity).The post-processed versions of these datasets are at http://www.cs.cmu.edu/~awm/kns

1. *Letter* originally had 26 classes: A-Z. We performed binary classification using the letter A as the positive class and "Not A" as negative.

2. *Movie* is a dataset from[11]. The TREC-2001 Video Track organized by NIST shot boundary Task. It is a 4 hours of video or 13 MPEG-1 video files at slightly over 2GB of data.

3. *Ipums* (from ipums.la.97). We predict *farm status*, which is binary.

4. *Kdd99(10%)* has a binary prediction: Normal vs. Attack.

Table 1: Datasets

| Dataset | Num. records | Num. Dimensions | Num. pos. | Dataset | Num. records | Num. Dimensions | Num. pos. |
|---|---|---|---|---|---|---|---|
| ds1 | 26733 | 6348 | 804 | ds1.10pca | 26733 | 10 | 804 |
| ds1.100pca | 26733 | 100 | 804 | ds2.100anchor | 88358 | 100 | 211 |
| ds2 | 88358 | 1100000 | 211 | J_Lee.100pca | 181395 | 100 | 299 |
| Letter | 20000 | 16 | 790 | Blanc_Mel | 186414 | 10 | 824 |
| Movie | 38943 | 62 | 7620 | Kdd99(10%) | 494021 | 176 | 97278 |
| Ipums | 70187 | 60 | 119 | | | | |

For each dataset, we tested $k = 9$ and $k = 101$. For KNS3, we used $q = \lceil k/2 \rceil$ when $k = 9$ and $q = \lceil pk/(n+p) \rceil$ when $k = 101$, where $p = $ Num.positive in the dataset and $n = $ Num.negative in the dataset. : a datapoint is classified as positive iff the majority of its $k$ nearest neighbors are positive. Each experiment performed 10-fold cross-validation. Thus, each experiment required $R$ $k$-NN classification queries (where $R$ is the number of records in the dataset) and each query involved the $k$-NN among $0.9R$ records. A naive implementation with no ball-trees would thus require $0.9R^2$ distance computations.These algorithms are all exact. No approximations were used in the classifications.

Table 2 shows the computational cost of naive $k$-NN, both in terms of the number of distance computations and the wall-clock time on an unloaded 2 GHz Pentium. We then examine the speedups of KNS1 (traditional use of Ball-trees) and our two new Ball-tree methods (KNS2 and KNS3). It is notable that for some high dimensional datasets, KNS1 does not produce an acceleration over naive. KNS2 and KNS3 do, however, and in some cases they are hundreds of times faster than KNS1. The *ds2* result is particularly interesting because it involves data in over a million dimensions. The first thing to notice is that conventional ball-trees (KNS1) were slightly *worse* than the naive $O(R^2)$ algorithm. In only one case was KNS2 inferior to naive and KNS3 was always superior. On some datasets KNS2 and KNS3 gave dramatic speedups.

Table 3 gives results for SVP1, the Ball-tree-based accelerator for SVM prediction[2] In general SVP1 appears to be 2-4 times faster than $SVM^{light}$[12], with two far more dramatic speedups in the case of two classification tasks where SVP1 quickly realizes that a large node near the top of its query tree can be pruned as negative. As with previous results, SVP1 is exact, and all predictions agree with SVM-Light. All these experiments used Radial Basis kernels, with kernel width tuned for optimal test-set performance.

Table 2: Number of distance computations and wall-clock-time for Naive *k*-NN classification (2nd column). Acceleration for normal use of ball-trees in col, 2 (in terms of num. distances and time). Accelerations of new methods KNS2 and KNS3 in other columns. Naive times are independent of *k*.

| | | NAIVE | | KNS1 | | KNS2 | | KNS3 | |
|---|---|---|---|---|---|---|---|---|---|
| | | dists | time (secs) | dists speedup | time speedup | dists speedup | time speedup | dists speedup | time speedup |
| ds1 | k=9 | $6.4 \times 10^8$ | 4830 | 1.6 | 1.0 | 4.7 | 3.1 | 12.8 | 5.8 |
| | k=101 | | | 1.0 | 0.7 | 1.6 | 1.1 | 10 | 4.2 |
| ds1.10pca | k=9 | $6.4 \times 10^8$ | 420 | 11.8 | 11.0 | 33.6 | 21.4 | 71 | 20 |
| | k=101 | | | 4.6 | 3.4 | 6.5 | 4.0 | 40 | 6.1 |
| ds1.100pca | k=9 | $6.4 \times 10^8$ | 2190 | 1.7 | 1.8 | 7.6 | 7.4 | 23.7 | 29.6 |
| | k=101 | | | 0.97 | 1.0 | 1.6 | 1.6 | 16.4 | 6.8 |
| ds2 | k=9 | $8.5 \times 10^9$ | 105500 | 0.64 | 0.24 | 14.0 | 2.8 | 25.6 | 3.0 |
| | k=101 | | | 0.61 | 0.24 | 2.4 | 0.83 | 28.7 | 3.3 |
| ds2.100- | k=9 | $7.0 \times 10^9$ | 24210 | 15.8 | 14.3 | 185.3 | 144 | 580 | 311 |
| | k=101 | | | 10.9 | 14.3 | 23.0 | 19.4 | 612 | 248 |
| J_Lee.100- | k=9 | $3.6 \times 10^{10}$ | 142000 | 2.6 | 2.4 | 28.4 | 27.2 | 15.6 | 12.6 |
| | k=101 | | | 2.2 | 1.9 | 12.6 | 11.6 | 37.4 | 27.2 |
| Blanc_Mel | k=9 | $3.8 \times 10^{10}$ | 44300 | 3.0 | 3.0 | 47.5 | 60.8 | 51.9 | 60.7 |
| | k=101 | | | 2.9 | 3.1 | 7.1 | 33 | 203 | 134.0 |
| Letter | k=9 | $3.6 \times 10^8$ | 290 | 8.5 | 7.1 | 42.9 | 26.4 | 94.2 | 25.5 |
| | k=101 | | | 3.5 | 2.6 | 9.0 | 5.7 | 45.9 | 9.4 |
| Movie | k=9 | $1.4 \times 10^9$ | 3100 | 16.1 | 13.8 | 29.8 | 24.8 | 50.5 | 22.4 |
| | k=101 | | | 9.1 | 7.7 | 10.5 | 8.1 | 33.3 | 11.6 |
| Ipums | k=9 | $4.4 \times 10^9$ | 9520 | 195 | 136 | 665 | 501 | 1003 | 515 |
| | k=101 | | | 69.1 | 50.4 | 144.6 | 121 | 5264 | 544 |
| Kddcup99 | k=9 | $2.7 \times 10^{11}$ | 1670000 | 4.2 | 4.2 | 574 | 702 | 4 | 4.1 |
| (10%) | k=101 | | | 4.2 | 4.2 | 187.7 | 226.2 | 3.9 | 3.9 |

Table 3: Comparison between SVM light and SVP1. We show the total number of distance computations made during the prediction phase for each method, and total wall-clock time.

| | SVM light distances | SVP1 distances | SVM light seconds | SVP1 seconds | speedup |
|---|---|---|---|---|---|
| ds1 | $6.4 \times 10^7$ | $1.8 \times 10^7$ | 394 | 171 | 2.3 |
| ds1.10pca | $6.4 \times 10^7$ | $1.8 \times 10^7$ | 60 | 23 | 2.6 |
| ds1.100pca | $6.4 \times 10^7$ | $2.3 \times 10^7$ | 259 | 92 | 2.8 |
| ds2.100pca | $7.0 \times 10^8$ | $1.4 \times 10^8$ | 2775 | 762 | 3.6 |
| J_Lee.100pca | $6.4 \times 10^6$ | $2 \times 10^6$ | 31 | 7 | 4.4 |
| Blanc_Mel | $1.2 \times 10^8$ | $3.6 \times 10^7$ | 61 | 26 | 2.3 |
| Letter | $2.6 \times 10^7$ | $1 \times 10^7$ | 21 | 11 | 1.9 |
| Ipums | $1.9 \times 10^8$ | $7.7 \times 10^4$ | 494 | 1 | 494 |
| Movie | $1.4 \times 10^8$ | $4.4 \times 10^7$ | 371 | 136 | 2.7 |
| Kddcup99(10%) | $6.3 \times 10^6$ | $2.8 \times 10^5$ | 69 | 1 | 69 |

## 4 Comments and related work

**Applicability of other proximity query work.** For the problem of "find the *k* nearest datapoints" (as opposed to our question of "perform *k*-NN or Kernel classification") in high dimensions, the frequent failure of traditional ball trees to beat naive has lead to some innovative alternatives, based on random projections, hashing discretized cubes, and acceptance of approximate answers. For example [7] gives a hashing method that was demonstrated to provide speedups over a ball-tree-based approach in 64 dimensions by a factor of 2-5 depending on how much error in the approximate answer was permitted. Another approximate *k*-NN idea is in [1], one of the first *k*-NN approaches to use a priority queue of nodes, in this case achieving a 3-fold speedup with an approximation to the true *k*-NN. However, these approaches are based on the notion that any points falling within a factor of $(1 + \varepsilon)$ times the true nearest neighbor distance are acceptable substitutes for the true nearest neighbor. Noting in particular that distances in high-dimensional spaces tend to occupy a decreasing range of continuous values [10], it remains an open question whether schemes based upon the absolute values of the distances rather than their *ranks* are relevant to the

classification task. Our approach, because it need not find the $k$-NN to answer the relevant statistical question, finds an answer without approximation. The fact that our methods are easily modified to allow $(1+\varepsilon)$ approximation in the manner of [1] suggests an obvious avenue for future research.

## Footnotes

[2]Because training SVMs is so expensive, some of the results below used reduced training sets.

# References

[1] S. Arya, D. Mount, N. Netanyahu, R. Silverman, and A. Wu. An optimal algorithm for approximate nearest neighbor searching fixed dimensions. *Journal of the ACM*, 45(6):891–923, 1998.

[2] S. D. Bay. UCI KDD Archive [http://kdd.ics.uci.edu]. Irvine, CA: University of California, Dept of Information and Computer Science, 1999.

[3] C. Burges. A tutorial on Support Vector Machines for Pattern Recognition. *Data Mining and Knowledge Discovery*, 2(2):955–974, 1998.

[4] P. Ciaccia, M. Patella, and P. Zezula. M-tree: An efficient access method for similarity search in metric spaces. In *Proceedings of the 23rd VLDB International Conference*, September 1997.

[5] K. Deng and A. W. Moore. Multiresolution Instance-based Learning. In *Proceedings of the Twelfth International Joint Conference on Artificial Intelligence*, pages 1233–1239, San Francisco, 1995. Morgan Kaufmann.

[6] J. H. Friedman, J. L. Bentley, and R. A. Finkel. An algorithm for finding best matches in logarithmic expected time. *ACM Transactions on Mathematical Software*, 3(3):209–226, September 1977.

[7] A. Gionis, P. Indyk, and R. Motwani. Similarity Search in High Dimensions via Hashing. In *Proc 25th VLDB Conference*, 1999.

[8] A. Gray and A. W. Moore. N-Body Problems in Statistical Learning. In Todd K. Leen, Thomas G. Dietterich, and Volker Tresp, editors, *Advances in Neural Information Processing Systems 13 (December 2000)*. MIT Press, 2001.

[9] A. Guttman. R-trees: A dynamic index structure for spatial searching. In *Proceedings of the Third ACM SIGACT-SIGMOD Symposium on Principles of Database Systems*. Assn for Computing Machinery, April 1984.

[10] J. M. Hammersley. The Distribution of Distances in a Hypersphere. *Annals of Mathematical Statistics*, 21:447–452, 1950.

[11] CMU informedia digital video library project. The trec-2001 video trackorganized by nist shot boundary task, 2001.

[12] T. Joachims. Making large-scale support vector machine learning practical. In A. Smola B. Schölkopf, C. Burges, editor, *Advances in Kernel Methods: Support Vector Machines*. MIT Press, Cambridge, MA, 1998.

[13] A. W. Moore. The Anchors Hierarchy: Using the Triangle Inequality to Survive High-Dimensional Data. In *Twelfth Conference on Uncertainty in Artificial Intelligence*. AAAI Press, 2000.

[14] S. M. Omohundro. Efficient Algorithms with Neural Network Behaviour. *Journal of Complex Systems*, 1(2):273–347, 1987.

[15] S. M. Omohundro. Bumptrees for Efficient Function, Constraint, and Classification Learning. In R. P. Lippmann, J. E. Moody, and D. S. Touretzky, editors, *Advances in Neural Information Processing Systems 3*. Morgan Kaufmann, 1991.

[16] D. Pelleg and A. W. Moore. Accelerating Exact $k$-means Algorithms with Geometric Reasoning. In *Proceedings of the Fifth International Conference on Knowledge Discovery and Data Mining*. ACM, 1999.

[17] F. P. Preparata and M. Shamos. *Computational Geometry*. Springer-Verlag, 1985.

[18] J. K. Uhlmann. Satisfying general proximity/similarity queries with metric trees. *Information Processing Letters*, 40:175–179, 1991.

[19] W. Zheng and A. Tropsha. A Novel Variable Selection QSAR Approach based on the K-Nearest Neighbor Principle. *J. Chem. Inf.Comput. Sci.*, 40(1):185–194, 2000.
